# Temporally changing synaptic plasticity

**Minija Tamosiunaite**[1,2]**, Bernd Porr**[3]**, and Florentin Wörgötter**[1,4]

[1] Department of Psychology, University of Stirling
Stirling FK9 4LA, Scotland
[2] Department of Informatics, Vytautas Magnus University
Kaunas, Lithuania
[3] Department of Electronics & Electrical Engineering, University of Glasgow
Glasgow, GT12 8LT, Scotland
[4] Bernstein Centre for Computational Neuroscience, University of Göttingen, Germany
{minija,worgott}@cn.stir.ac.uk; b.porr@elec.gla.ac.uk

## Abstract

Recent experimental results suggest that dendritic and back-propagating spikes can influence synaptic plasticity in different ways [1]. In this study we investigate how these signals could temporally interact at dendrites leading to changing plasticity properties at local synapse clusters. Similar to a previous study [2], we employ a differential Hebbian plasticity rule to emulate spike-timing dependent plasticity. We use dendritic (D-) and back-propagating (BP-) spikes as post-synaptic signals in the learning rule and investigate how their interaction will influence plasticity. We will analyze a situation where synapse plasticity characteristics change in the course of time, depending on the type of post-synaptic activity momentarily elicited. Starting with weak synapses, which only elicit local D-spikes, a slow, unspecific growth process is induced. As soon as the soma begins to spike this process is replaced by fast synaptic changes as the consequence of the much stronger and sharper BP-spike, which now dominates the plasticity rule. This way a winner-take-all-mechanism emerges in a two-stage process, enhancing the best-correlated inputs. These results suggest that synaptic plasticity is a temporal changing process by which the computational properties of dendrites or complete neurons can be substantially augmented.

## 1  Introduction

The traditional view on Hebbian plasticity is that the correlation between pre- and postsynaptic events will drive learning. This view ignores the fact that synaptic plasticity is driven by a whole sequence of events and that some of these events are causally related. For example, usually through the synaptic activity at a cluster of synapses the postsynaptic spike will be triggered. This signal can then travel retrogradely into the dendrite (as a so-called back-propagating- or BP-spike, [3]), leading to a depolarization at this and other clusters of synapses by which their plasticity will be influenced. More locally, something similar can happen if a cluster of synapses is able to elicit a dendritic spike (D-spike, [4, 5]), which may not travel far, but which certainly leads to a local depolarization "under" these and

adjacent synapses, triggering synaptic plasticity of one kind or another. Hence synaptic plasticity seems to be to some degree influenced by recurrent processes. In this study, we will use a differential Hebbian learning rule [2, 6] to emulate spike timing dependent plasticity (STDP, [7, 8]). With one specifically chosen example architecture we will investigate how the temporal relation between dendritic- and back propagating spikes could influence plasticity. Specifically we will report how learning could change *during* the course of network development, and how that could enrich the computational properties of the affected neuronal compartments.

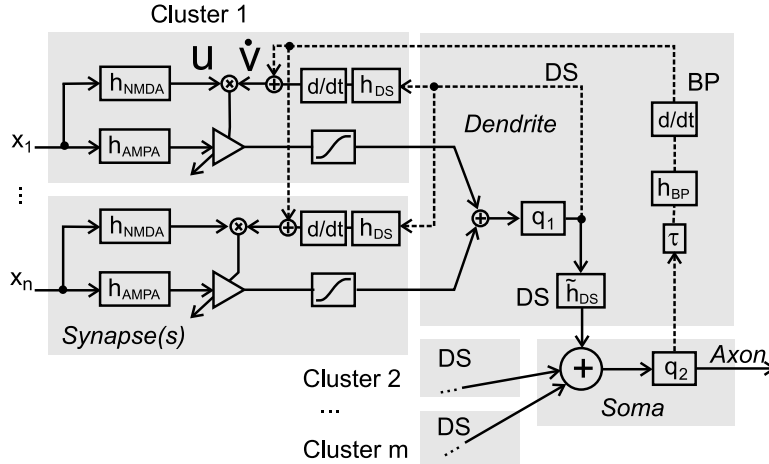

Figure 1: Basic learning scheme with $x_1, ..., x_n$ representing inputs to cluster 1, $h_{AMPA}$, $h_{NMDA}$ - filters shaping AMPA and NMDA signals, $h_{DS}$, $\tilde{h}_{DS}$, $h_{BP}$ - filters shaping D and BP-spikes, $q_1$, $q_2$ - differential thresholds, $\tau$ - a delay. Weight impact is saturated. Only the first of $m$ clusters is shown explicitly; clusters $2, 3, ..., m$ would be employing the same BP spike (not shown). The symbol $\oplus$ represents a summation node and $\otimes$ multiplication.

## 2 The Model

A block diagram of the model is shown in Fig. 1. The model includes several clusters of synapses located on dendritic branches. Dendritic spikes are elicited following the summation of several AMPA signals passing threshold $q_1$. NMDA receptor influence on dendritic spike generation was not considered as the contribution of NMDA potentials to the total membrane potential is substantially smaller than that of AMPA channels at a mixed synapse.

Inputs to the model arrive in groups, but each input line gets only one pulse in a given group (Fig. 2 C). Each synaptic cluster is limited to generating one dendritic spike from one arriving pulse group. Cell firing is not explicitly modelled but said to be achieved when the summation of several dendritic spikes at the cell soma has passed threshold $q_2$. This leads to a BP-spike. Progression of signals along a dendrite is not modelled explicitly, but expressed by means of delays. Since we do not model biophysical processes, all signal *shapes* are obtained by appropriate filters $h$, where $u = x * h$ is the convolution of spike train $x$ with filter $h$.

A differential Hebbian-type learning rule is used to drive synaptic plasticity [2, 6] with $\dot{\rho} = \mu u \dot{v}$, where $\rho$ denotes synaptic weight, $u$ stands for the synaptic input, $v$ for the output, and $\mu$ for the learning rate. see e.g.; $u$ and $\dot{v}$ annotations in Fig. 1, top left.

NMDA signals are used as the pre-synaptic signals, dendritic spikes, or dendritic spikes complemented by back-propagating spikes, define the post-synaptic signals for the learning rule. In addition, synaptic weights were sigmoidally saturated with limits zero and one. Filter shapes forming AMPA and NMDA channel responses, as well as back- propagating spikes and some forms of dendritic spikes used in this study were described by:

$$h(t) = \frac{e^{-2\pi t/\tau} - e^{-8\pi t/\tau}}{6\pi/\tau} \qquad (1)$$

where $\tau$ determines the total duration of the pulse. The ratio between rise and fall time is $1 : 4$. We use for AMPA channels: $\tau = 6\ ms$, for NMDA channels: $\tau = 120\ ms$, for dendritic spikes: $\tau = 235\ ms$, and for BP-spikes: $\tau = 40\ ms$.

Note, we are approximating the NMDA characteristic by a non-voltage dependent filter function. In conjunction with STDP, this simplification is justified by Saudargiene et al [2, 9], showing that voltage dependency induces only a second-order effect on the shape of the STDP curve.

Individual input timings are drawn from a uniform distribution from within a pre-specified interval which can vary under different conditions. We distinguish three basic input groups: *strongly correlated* inputs (several inputs over an interval of up to 10 ms), *less correlated* (dispersed over an interval of 10-100 ms) and *uncorrelated* (dispersed over the interval of more than 100 ms).

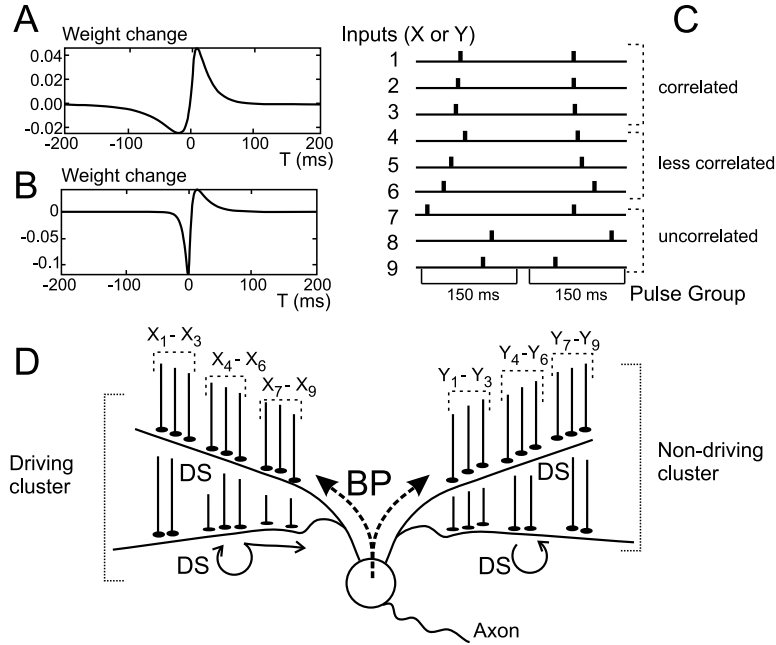

Figure 2: Example STDP curves (A,B), input pulse distribution (C), and model setup (D). A) STDP curve obtained with a D-spike using Eq. 1 with $\tau = 235\ ms$, B) from a BP spike with $\tau = 40\ ms$. C) Example input pulse distribution for two pulse groups. D) Model neuron with two dendritic branches (left and right), consisting of two sub-branches which get inputs $X$ or $Y$, which are similar for either side. DS stands for D-spike, BP for a BP-spike.

# 3  Results

## 3.1  Experimental setup

Fig. 2 A,B shows two STDP curves, one obtained with a wide D-spike the other one with a much sharper BP-spike. The study investigates interactions of such post-synaptic signals in time. Though the signals interact linearly, the much stronger BP signal dominates learning when elicited. In the absence of a BP spike the D-spike dominates plasticity. This seems to correspond to new physiological observations concerning the relations between post-synaptic signals and the actually expressed form of plasticity [10]. We specifically investigate a two-phase processes, where plasticity is *first* dominated by the D- spike and *later* by a BP-spike.

Fig. 2 D shows a setup in which two-phase plasticity could arise. We assume that inputs to compact clusters of synapses are similar (e.g. all left branches in Fig. 2 D) but dissimilar over larger distances (between left and right branches). First, e.g. early in development, synapses may be weak and only the conjoint action of many synchronous inputs will lead to a local D-spike. Local plasticity from these few D-spikes (indicated by the circular arrow under the dendritic branches in Fig. 2) strengthens these synapses and at some point D-spikes are elicited more reliably at conjoint branches. This could finally also lead to spiking at the soma and, hence, to a BP-spike, changing plasticity of the individual synapses.

To emulate such a multi-cluster system we actually model only one left and one right branch. Plasticity in both branches is driven by D-spikes in the first part of the experiment. Assuming that at some point the cell will be driven into spiking, a BP-spike is added after several hundred pulse groups (second part of the experiment).

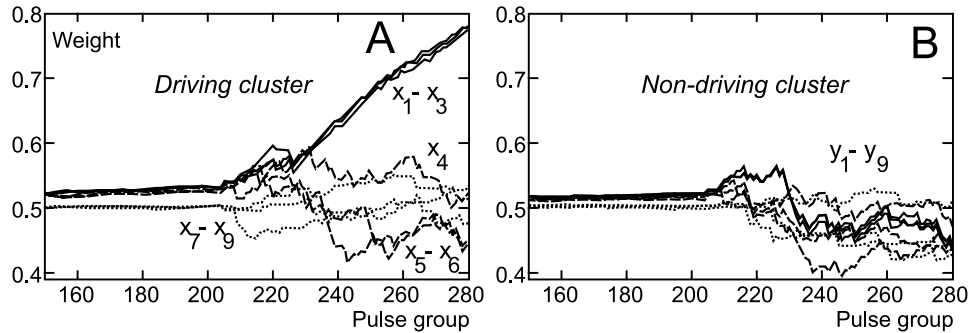

Figure 3: Temporal weight development for the setup shown in Fig 2 with one sub-branch for the driving cluster (A), and one for the non-driving cluster (B). Initially all weights grow gradually until the driving cluster leads to a BP-spike after 200 pulse groups. Thus only the weights of its group $x_1 - x_3$ will continue to grow, now at an increased rate.

## 3.2  An emerging winner-take-all mechanism

In Fig. 3 we have simulated two clusters each with nine synapses. For both clusters, we assume that the input activity for three synapses is closely correlated and that they occur in a temporal interval of $6\ ms$ (group $x, y$: $1 - 3$). Three other inputs are wider dispersed (interval of $35\ ms$, group $x, y$: $4 - 6$) and the three remaining ones arrive uncorrelated in an interval of $150\ ms$ (group $x, y$: $7 - 9$). The activity of the second cluster is determined by the same parameters. Pulse groups arriving at the second cluster, however, were randomly shifted by maximally $\pm 20\ ms$ relative to the centre of the pulse group of the first cluster.

All synapses start with weights 0.5, which will not suffice to drive the soma of the cell into spiking. Hence initially plasticity can only take place by D-spikes, and we assume that D-spikes will not reach the other cluster. Hence, learning is local. The wide D-spike leads to a broad learning curve which has a span of about $\pm 17.5 ms$ around zero, covering the dispersion of input groups $1 - 3$ as well as $4 - 6$. Furthermore it has a slightly bigger area under the LTP part as compared to the LTD part. As a consequence, in both diagrams (Fig. 3 A,B) we see that *all* weights $1 - 6$ grow, only for the least correlated input $6 - 9$ the weights remain close their origin. The correlated group $1 - 3$, however, benefits most strongly, because it is more likely that a D-spike will be elicited by this group than by any other combination.

Conjoint growth at a whole cluster of such synapses would at some point drive the cell into somatic firing. Here we just assume that this happens for one cluster (Fig. 3 A) at a certain time point. This can, for example, be the case when the input properties of the two input groups are different leading to (slightly) less weight growth in the other cluster. As soon as this happens a BP-spike is triggered and the STDP curve takes a narrow shape similar to that in Fig. 2 B now strongly enhancing all causally driving synapses, hence group $x_1 - x_3$ (Fig. 3 A). This group grows at an increased rate while all other synapses shrink. Hence, in general this system exhibits two-phase plasticity. This result was reproduced in a model with 100 synapses in each input group (data not shown) and in the next sections we will show that a system with two growth phases is rather robust against parameter variations.

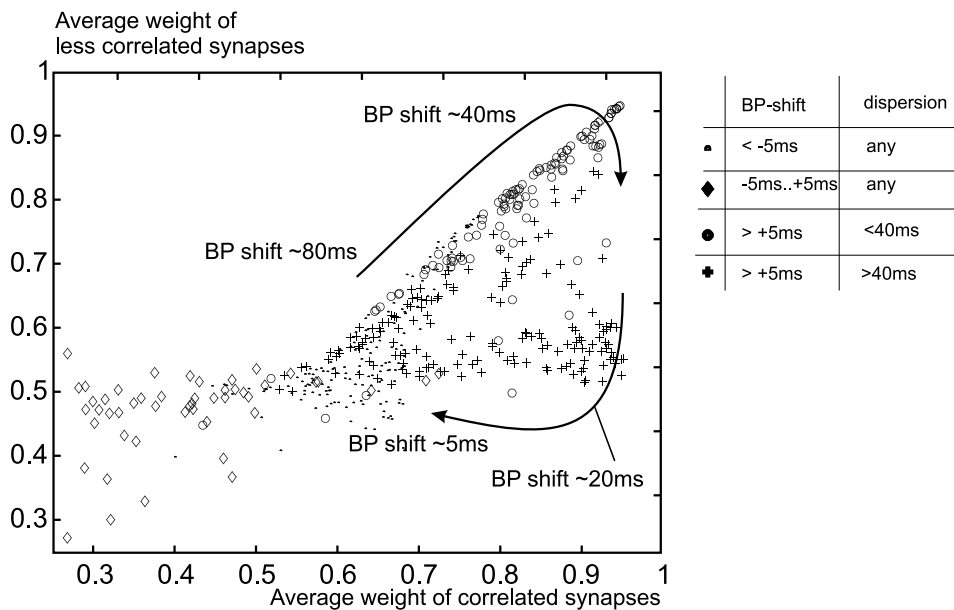

Figure 4: Robustness of the observed effects. Plotted are the average weights of the less correlated group (ordinate) against the correlated group (abscissa). Simulation with three correlated and three less correlated inputs, for AMPA: $\tau = 6\ ms$, for NMDA: $\tau = 117\ ms$, for D-spike: $\tau = 235\ ms$, for BP-spike: $\tau = 6 - 66\ ms$, $q_1 = 0.14$. D/BP spike amplitude relation from $1/1.5$ to $1/15$, depending on BP-spike width, and keeping the area under the BP-spike constant, $\mu = 0.2$. For further explanation see text.

### 3.3 Robustness

This system is not readily suited for analytical investigation like the simpler ones in [9]. However, a fairly exhaustive parameter analysis is performed. Fig. 4 shows a plot of 350 experiments with the same basic architecture, using only one synapse cluster and the same chain of events as before but with different parameter settings. Only "strong correlated" ($< 10\ ms$) and "less correlated" ($10 - 100\ ms$) inputs were used in this experiment. Each point represents one experiment consisting of 600 pulse groups. On the abscissa we plot the average weight of the three correlated synapses; on the ordinate the average weight of the three less correlated synapses after these 600 pulse groups. We assume, as in the last experiment, that a BP-spike is triggered as soon as $q_2$ is passed, which happens around pulse group 200 in all cases.

Four parameters were varied to obtain this plot. **(1)** The width of the BP-spike was varied between $5\ ms$ and $50\ ms$. **(2)** The interval width for the temporal distribution of the three correlated spikes was varied between $1\ ms$ and $10\ ms$. Hence $1\ ms$ amounts to three synchronously elicited spikes. **(3)** The interval width for the temporal distribution of the three less correlated spikes was varied between $1\ ms$ and $100\ ms$. **(4)** The shift of the BP-spike with respect to the beginning of the D-spike was varied in an interval of $\pm 80\ ms$.

Mainly parameters 3 and 4 have an effect on the results. The first parameter, BP spike width, shows some small interference with the spike shift for the widest spikes. The second parameter has almost no influence, due to the small parameter range ($10\ ms$). Symbol coding is used in Fig. 4 to better depict the influence of parameters 3 and 4 in their different ranges. Symbols "dots", "diamonds" and "others" (circles and plusses) refer to a BP-spike shifts: of less than $-5\ ms$ (dots), between $-5\ ms$ and $+5\ ms$ (diamonds) and larger than $+5\ ms$ (circles and pluses). Circles in the latter region show cases with the less correlated dispersion interval below $40\ ms$, and plusses the cases of the dispersion $40\ ms$ or higher. The "dot" region ($-5\ ms$) shows cases where correlated synapses will grow, while less correlated synapses can grow or shrink. This happens because the BP spike is too early to influence plasticity in the strongly correlated group, which will grow by the DS-mechanism only, but the BP-spike still falls in the dispersion range of the less correlated group, influencing its weights. At a shift of $-5\ ms$ a fast transition in the weight development occurs. The reason for this transition is that the BP-spike, being very close to the D-spike, overrules the effect of the D-spike. The randomness whether the input falls into pre- or post-output zone in both, correlated and less correlated, groups is large enough, and leads to weights staying close to origin or to shrinkage. The circles and plusses encode the dispersion of the wide, less correlated spike distributions in the case when time shifts of the BP-spike are positive ($> 5\ ms$, hence BP-spike after D-spike). Dispersions are getting wider essentially from top to bottom (circle to dot). Clearly this shows that there are many cases corresponding to the example depicted in Fig. 3 (horizontal tail of Fig. 4 A), but there are also many conventional situations, where both weight-groups just grow in a similar way (diagonal).

The data points show a certain regularity when the BP spike shift moves from big values towards the borderline of $+5\ ms$, where the weights stop to grow. For big shifts, points cluster on the upper, diagonal tail in or near the dot region. With a smaller BP spike shift points move up this tail and then drop down to the horizontal tail, which occurs for shifts of about $20\ ms$. This pattern is typical for the bigger dispersion in the range of $20 - 60\ ms$ and data points essentially follow the circle drawn in the figure.

This happens because as soon as the BP-spike gets closer to the D-spike, it will start to exert its influence. But this will first only affect the less correlated group as there are almost always some inputs so late that they "collide" with the BP-spike. Time of collision, however, is random and sometimes these input are "pre" while sometimes they are "post" with respect to the BP-spike. Hence LTP and LTD will be essentially balanced in the less

correlated group, leading on average to zero weight growth. This effect is most pronounced when the less correlated group has an intermediate dispersion (see the circles from the upper tail dropping to the lower tail in the range of dispersions $20 - 40\ ms$ ), while it does not occur if the dispersion of correlated and less correlated groups are similar ($1 - 20\ ms$).

Furthermore, the clear separation into the top- (circles, $1 - 40\ ms$) and bottom-tail (plusses, $61 - 100\ ms$) indicates that it is possible to let the parameters drift quite a bit without leaving the respective regions. Hence, while the moment-to-moment weight growth might change, the general pattern will stay the same.

## 4   Discussion

Just like with the famous Baron von Münchausen, who was able to pull himself out of a swamp by his own hair, the current study suggests that plasticity change as a consequence of itself might lead to specific functional properties. In order to arrive at this conclusion, we have used a simplified model of STDP and combined it with a custom designed and also simplified dendritic architecture. Hence, can the conclusions of this study be valid and where are the limitations? We believe that answer to the first question is affirmative because the degree of abstraction used in this model and the complexity of the results match. This model never attempted to address the difficult issues of the biophysics of synaptic plasticity (for a discussion see [2]) and it was also not our goal to investigate the mechanisms of signal propagation in a dendrite [11]. Both aspects had been reduced to a few basic descriptors and this way we were able to show for the first time that a useful synaptic selection process can develop over time. The system consisted of a first "pre-growth" phase (until the BP-spike sets in) followed by a second phase where only one group of synapses grows strongly, while the others shrink again. In general this example describes a scenario where groups of synapses first undergo less selective classical Hebbian-like growth, while later more pronounced STDP sets in, selecting only the main driving group. We believe that in the early development of a real brain such a two-phase system might be beneficial for the stable selection of those synapses that are better correlated. It is conceivable that at early developmental stages correlations are in general weaker, while the number of inputs to a cell is probably much higher than in the adult stage, where many have been pruned. Hence highly selective and strong STDP-like plasticity employed too early might lead to a noise-induced growth of "the wrong" synapses. This, however, might be prevented by just such a soft pre-selection mechanisms which would gradually drive clusters of synapses apart by a local dendritic process before the stronger influence of the back-propagating spike sets in. This is supported by recent results from Holthoff et al [1, 12], who have shown that D-spikes will lead to a different type of plasticity than BP-spikes in layer 5 pyramidal cells in mouse cortex. Many more complications exist, for example the assumed chain of events of D- and BP-spikes may be very different in different neurons and the interactions between these signals may be far more non-linear (but see [10]). This will require to re-address these issues in greater detail when dealing with a specific given neuron but the general conclusions about the self-influencing and local [2, 13] character of synaptic plasticity and their possible functional use should hopefully remain valid.

## 5   Acknowledgements

The authors acknowledge the support from SHEFC INCITE and IBRO. We are grateful to B. Graham, L. Smith and D. Sterratt for their helpful comments on this work. The authors wish to especially express their thanks to A. Saudargiene for her help at many stages in this project.

# References

[1] K. Holthoff, Y. Kovalchuk, R. Yuste, and A. Konnerth. Single-shock plasticity induced by local dendritic spikes. In *Proc. Göttingen NWG Conference*, page 245B, 2005.

[2] A. Saudargiene, B. Porr, and F. Wörgötter. How the shape of pre- and postsynaptic signals can influence STDP: a biophysical model. *Neural Comp.*, 16:595–626, 2004.

[3] N.L. Golding, W. L. Kath, and N. Spruston. Dichotomy of action-potential backpropagation in ca1 pyramidal neuron dendrites. *J Neurophysiol.*, 86:2998–3010, 2001.

[4] M. E. Larkum, J. J. Zhu, and B. Sakmann. Dendritic mechanisms underlying the coupling of the dendritic with the axonal action potential initiation zone of adult rat layer 5 pyramidal neurons. *J. Physiol. (Lond. )*, 533:447–466, 2001.

[5] N. L. Golding, P. N. Staff, and N. Spurston. Dendritic spikes as a mechanism for cooperative long-term potentiation. *Nature*, 418:326–331, 2002.

[6] B. Porr and F. Wörgötter. Isotropic sequence order learning. *Neural Comp.*, 15:831–864, 2003.

[7] J. C. Magee and D. Johnston. A synaptically controlled, associative signal for Hebbian plasticity in hippocampal neurons. *Science*, 275:209–213, 1997.

[8] H. Markram, J. Lübke, M. Frotscher, and B. Sakmann. Regulation of synaptic efficacy by coincidence of postsynaptic APs and EPSPs. *Science*, 275:213–215, 1997.

[9] A. Saudargiene, B. Porr, and F. Wörgötter. Local learning rules: predicted influence of dendritic location on synaptic modification in spike-timing-dependent plasticity. *Biol. Cybern.*, 92:128–138, 2005.

[10] H.-X. Wang, Gerkin R. C., D. W. Nauen, and G.-Q. Bi. Coactivation and timing-dependent integration of synaptic potentiation and depression. *Nature Neurosci.*, 8:187–193, 2005.

[11] P. Vetter, A. Roth, and M. Häusser. Propagation of action potentials in dendrites depends on dendritic morphology. *J. Neurophsiol.*, 85:926–937, 2001.

[12] K. Holthoff, Y. Kovalchuk, R. Yuste, and A. Konnerth. Single-shock LTD by local dendritic spikes in pyramidal neurons of mouse visual cortex. *J. Physiol.*, 560.1:27–36, 2004.

[13] R. C. Froemke, M-m. Poo, and Y. Dan. Spike-timing-dependent synaptic plasticity depends on dendritic location. *Nature*, 434:221–225, 2005.
